# Partially Observed Maximum Entropy Discrimination Markov Networks

**Jun Zhu**[†]          **Eric P. Xing**[‡]          **Bo Zhang**[†]

[†]State Key Lab of Intelligent Tech & Sys, Tsinghua National TNList Lab, Dept. Comp Sci & Tech,
[†]Tsinghua University, Beijing China. `jun-zhu@mails.thu.edu.cn`; `dcszb@thu.edu.cn`
[‡]School of Comp. Sci., Carnegie Mellon University, Pittsburgh, PA 15213, `epxing@cs.cmu.edu`

## Abstract

Learning graphical models with hidden variables can offer semantic insights to complex data and lead to salient structured predictors without relying on expensive, sometime unattainable fully annotated training data. While likelihood-based methods have been extensively explored, to our knowledge, learning structured prediction models with latent variables based on the max-margin principle remains largely an open problem. In this paper, we present a partially observed Maximum Entropy Discrimination Markov Network (PoMEN) model that attempts to combine the advantages of Bayesian and margin based paradigms for learning Markov networks from partially labeled data. PoMEN leads to an averaging prediction rule that resembles a Bayes predictor that is more robust to overfitting, but is also built on the desirable discriminative laws resemble those of the $M^3N$. We develop an EM-style algorithm utilizing existing convex optimization algorithms for $M^3N$ as a subroutine. We demonstrate competent performance of PoMEN over existing methods on a real-world web data extraction task.

## 1    Introduction

Inferring structured predictions based on high-dimensional, often multi-modal and hybrid covariates remains a central problem in data mining (e.g., web-info extraction), machine intelligence (e.g., machine translation), and scientific discovery (e.g., genome annotation). Several recent approaches to this problem are based on learning discriminative graphical models defined on composite features that explicitly exploit the structured dependencies among input elements and structured interpretational outputs. Different learning paradigms have been explored, including the maximum conditional likelihood [7] and max-margin learning [2, 12, 13], with remarkable success.

However, the problem of structured input/output learning can be intriguing and significantly more difficult when there exist hidden substructures in the data, which is not uncommon in realistic problems. As is well-known in the probabilistic graphical model literature, hidden variables can facilitate natural incorporation of structured domain knowledge such as latent semantic concepts or unobserved dependence hierarchies into the model, which can often result in more intuitive representation and more compact parameterization of the model; but learning a partially observed model is often non-trivial because it involves optimizing against a more complex cost function, which is usually not convex and requires additional efforts to impute or marginalize out hidden variables. Most existing work along this line, such as the hidden CRF for object recognition [9] and scene segmentation [14] and the dynamic hierarchical MRF for web data extraction [18], falls in the likelihood-based learning. For the max-margin learning, which is arguably a more desirable discriminative learning paradigm in many application scenarios, learning a Makov network with hidden variables can be extremely difficult and little work has been done except [11], where, in order to obtain a convex program, the uncertainty in mixture modeling is simplified by a reduction using the MAP component.

A major reason for the difficulty of considering latent structures in max-margin models is the lack of a natural probabilistic interpretation of such models, which on the other hand offers the key insight in likelihood-based learning to design algorithms such as EM for learning partially observed models. Recent work on semi-supervised or unsupervised max-margin learning [1, 4, 16] was all short of an explicit probabilistic interpretation of their algorithms of handling latent variables. The recently proposed *Maximum Entropy Discrimination Markov Networks* (MaxEnDNet) [20, 19] represent a key advance in this direction. MaxEnDNet offers a general framework to combine Bayesian-style learning and max-margin learning in structured prediction. Given a prior distribution of a structured-prediction model, and leveraging a new prediction-rule that is based on a weighted average over an ensemble of prediction models, MaxEnDNet adopts a *structured minimum relative entropy* principle to learn a posterior distribution of the prediction model in a subspace defined by a set of *expected* margin constraints. This elegant combination of probabilistic and maximum margin concepts provides a natural path to incorporate hidden structured variables in learning max-margin Markov networks ($M^3N$), which is the focus of this paper.

It has been shown in [20] that, in the fully observed case, MaxEnDNet subsumes the standard $M^3N$ [12]. But MaxEnDNet in its full generality offers a number of important advantages while retaining all the merits of the $M^3N$. For example, structured prediction under MaxEnDNet is based on an averaging model and therefore enjoys a desirable smoothing effect, with a uniform convergence bound on generalization error, as shown in [20]; MaxEnDNet admits a prior that can be designed to introduce useful regularization effects, such as a sparsity bias, as explored in the Laplace $M^3N$ [19, 20]. In this paper, we explore yet another advantage of MaxEnDNet stemmed from the Bayesian-style max-margin learning formalism on incorporating hidden variables. We present the *partially observed* MaxEnDNet (PoMEN), which offers a principled way to incorporate latent structures carrying domain knowledge and learn a discriminative model with partially labeled data. The reducibility of MaxEnDNet to $M^3N$ renders many existing convex optimization algorithms developed for learning $M^3N$ directly applicable as subroutines for learning our proposed model. We describe an EM-style algorithm for PoMEN based on existing algorithms for $M^3N$. As a practical application, we apply the proposed model to a web data extraction task–product information extraction, where collecting fully labeled training data is very difficult. The results show the promise of max-margin learning as opposed to likelihood-based estimation in the presence of hidden variables.

The paper is organized as follows. Section 2 reviews the basic max-margin structured prediction formalism and MaxEnDNet. Section 3 presents the partially observed MaxEnDNet. Section 4 applies the model to real web data extraction, and Section 5 brings this paper to a conclusion.

## 2  Preliminaries

Our goal is to learn a predictive function $h : \mathcal{X} \mapsto \mathcal{Y}$ from a structured input $\mathbf{x} \in \mathcal{X}$ to a structured output $\mathbf{y} \in \mathcal{Y}$, where $\mathcal{Y} = \mathcal{Y}_1 \times \cdots \times \mathcal{Y}_l$ represents a combinatorial space of structured interpretations of multi-facet objects. For example, in part-of-speech (POS) tagging, $\mathcal{Y}_i$ consists of all the POS tags and each label $\mathbf{y} = (y_1, \cdots, y_l)$ is a sequence of POS tags, and each input $\mathbf{x}$ is a sentence (word sequence). We assume that the feasible set of labels $\mathcal{Y}(\mathbf{x}) \subseteq \mathcal{Y}$ is finite for any $\mathbf{x}$.

Let $F(\mathbf{x}, \mathbf{y}; \mathbf{w})$ be a parametric discriminant function. A common choice of $F$ is a linear model, where $F$ is defined by a set of $K$ feature functions $f_k : \mathcal{X} \times \mathcal{Y} \mapsto \mathbb{R}$ and their weights $w_k$: $F(\mathbf{x}, \mathbf{y}; \mathbf{w}) = \mathbf{w}^\top \mathbf{f}(\mathbf{x}, \mathbf{y})$. A commonly used predictive function is:

$$h_0(\mathbf{x}; \mathbf{w}) = \arg \max_{\mathbf{y} \in \mathcal{Y}(\mathbf{x})} F(\mathbf{x}, \mathbf{y}; \mathbf{w}). \tag{1}$$

By using different loss functions, the parameters $\mathbf{w}$ can be estimated by maximizing the conditional likelihood [7] or by maximizing the margin [2, 12, 13] on labeled training data.

### 2.1  Maximum margin Markov networks

Under the $M^3N$ formalism, which we will generalize in this paper, given a set of fully labeled training data $\mathcal{D} = \{(\mathbf{x}^i, \mathbf{y}^i)\}_{i=1}^N$, the max-margin learning [12] solves the following optimization problem and achieves an optimum point estimate of the weight vector $\mathbf{w}$:

$$\text{P0 } (M^3N): \qquad \min_{\mathbf{w} \in \mathcal{F}_0, \xi \in \mathbb{R}_+^N} \frac{1}{2}\|\mathbf{w}\|^2 + C \sum_{i=1}^N \xi_i, \tag{2}$$

where $\xi_i$ represents a slack variable absorbing errors in training data, $C$ is a positive constant, $\mathbb{R}_+$ denotes non-negative real numbers, and $\mathcal{F}_0$ is the feasible space for $\mathbf{w}$: $\mathcal{F}_0 = \{\mathbf{w} : \mathbf{w}^\top \Delta \mathbf{f}_i(\mathbf{y}) \geq$

$\Delta \ell_i(\mathbf{y}) - \xi_i; \ \forall i, \forall \mathbf{y} \neq \mathbf{y}^i \}$, of which $\Delta \mathbf{f}_i(\mathbf{y}) = \mathbf{f}(\mathbf{x}^i, \mathbf{y}^i) - \mathbf{f}(\mathbf{x}^i, \mathbf{y})$, $\mathbf{w}^\top \Delta \mathbf{f}_i(\mathbf{y})$ is the "margin" between the true label $\mathbf{y}^i$ and a prediction $\mathbf{y}$, and $\Delta \ell_i(\mathbf{y})$ is a loss function with respect to $\mathbf{y}^i$.

Various loss functions have been proposed for P0. In this paper, we adopt the *hamming loss* [12]: $\Delta \ell_i(\mathbf{y}) = \sum_{j=1}^{|\mathbf{x}^i|} \mathbb{I}(y_j \neq y_j^i)$, where $\mathbb{I}(\cdot)$ is an indicator function that equals to 1 if the argument is true and 0 otherwise. The optimization problem P0 is intractable because of the exponential number of constraints in $\mathcal{F}_0$. Exploring sparse dependencies among individual labels $y_i$ in $\mathbf{y}$, as reflected in the specific design of the feature functions (e.g., based on pair-wise labeling potentials), efficient optimization algorithms based on cutting-plane [13] or message-passing [12], and various gradient-based methods [3, 10] have been proposed to obtain approximate solution to P0. As described shortly, these algorithms can be directly employed as subroutines in solving our proposed model.

## 2.2 Maximum Entropy Discrimination Markov Networks

Instead of predicting based on a single rule $F(\cdot; \mathbf{w})$ as in M³N using $\mathbf{w}$, the *structured maximum entropy discrimination* formalism [19] facilitates a Bayes-style prediction by averaging $F(\cdot; \mathbf{w})$ over a distribution of rules according to a posterior distribution of the weights, $p(\mathbf{w})$:

$$h_1(\mathbf{x}) = \arg \max_{\mathbf{y} \in \mathcal{Y}(\mathbf{x})} \int p(\mathbf{w}) F(\mathbf{x}, \mathbf{y}; \mathbf{w}) \, \mathrm{d}\mathbf{w}, \tag{3}$$

where $p(\mathbf{w})$ is learned by solving an optimization problem referred to as a *maximum entropy discrimination Markov network* (MaxEnDNet, or MEN) [20] that elegantly combines Bayesian-style learning with max-margin learning. In a MaxEnDNet, a prior over $\mathbf{w}$ is introduced to regularize its distribution, and the margins resulting from predictor (3) are used to define a feasible distribution subspace. More formally, given a set of fully observed training data $\mathcal{D}$ and a prior distribution $p_0(\mathbf{w})$, MaxEnDNet solves the following problem for an optimal posterior $p(\mathbf{w}|\mathcal{D})$ or $p(\mathbf{w})$:

$$\text{P1 (MaxEnDNet)}: \quad \min_{p(\mathbf{w}) \in \mathcal{F}_1, \xi \in \mathbb{R}_+^N} KL(p(\mathbf{w}) || p_0(\mathbf{w})) + U(\xi), \tag{4}$$

where the objective function $KL(p(\mathbf{w})||p_0(\mathbf{w})) + U(\xi)$ is known as the generalized entropy [8, 5], or regularized KL-divergence, and $U(\xi)$ is a closed proper convex function over the slack variables $\xi$. $U$ is also known as an additional "potential" term in the maximum entropy principle. The feasible distribution subspace $\mathcal{F}_1$ is defined as follows:

$$\mathcal{F}_1 = \Big\{ p(\mathbf{w}) : \int p(\mathbf{w})[\Delta F_i(\mathbf{y}; \mathbf{w}) - \Delta \ell_i(\mathbf{y})] \, \mathrm{d}\mathbf{w} \geq -\xi_i, \ \forall i, \ \forall \mathbf{y} \Big\},$$

where $\Delta F_i(\mathbf{y}; \mathbf{w}) = F(\mathbf{x}^i, \mathbf{y}^i; \mathbf{w}) - F(\mathbf{x}^i, \mathbf{y}; \mathbf{w})$.

P1 is a variational optimization problem over $p(\mathbf{w})$ in the feasible subspace $\mathcal{F}_1$. Since both the KL-divergence and the $U$ function in P1 are convex, and the constraints in $\mathcal{F}_1$ are linear, P1 is a convex program. Thus, one can apply the calculus of variations to the Lagrangian to obtain a variational extremum, followed by a dual transformation of P1. As proved in [20], solution to P1 leads to a GLIM for $p(\mathbf{w})$, whose parameters are closely connected to the solution of the M³N.

**Theorem 1 (MaxEnDNet (adapted from [20]))** *The variational optimization problem P1 underlying a MaxEnDNet gives rise to the following optimum distribution of Markov network parameters:*

$$p(\mathbf{w}) = \frac{1}{Z(\alpha)} p_0(\mathbf{w}) \exp \Big\{ \sum_{i, \mathbf{y}} \alpha_i(\mathbf{y})[\Delta F_i(\mathbf{y}; \mathbf{w}) - \Delta \ell_i(\mathbf{y})] \Big\}, \tag{5}$$

*where $Z(\alpha)$ is a normalization factor and the Lagrangian multipliers $\alpha_i(\mathbf{y})$ (corresponding to constraints in $\mathcal{F}_1$) can be obtained by solving the following dual problem of P1:*

$$\text{D1}: \quad \max_{\alpha} \ -\log Z(\alpha) - U^\star(\alpha)$$

$$\text{s.t.} \ \alpha_i(\mathbf{y}) \geq 0, \ \forall i, \ \forall \mathbf{y},$$

*where $U^\star(\cdot)$ is the conjugate of the slack function $U(\cdot)$, i.e., $U^\star(\alpha) = \sup_\xi \big( \sum_{i,\mathbf{y}} \alpha_i(\mathbf{y})\xi_i - U(\xi) \big)$.*

It can be shown that when $F(\mathbf{x}, \mathbf{y}; \mathbf{w}) = \mathbf{w}^\top \mathbf{f}(\mathbf{x}, \mathbf{y})$, $U(\xi) = C \sum_i \xi_i$, and $p_0(\mathbf{w})$ is a standard Gaussian $\mathcal{N}(\mathbf{w}|0, I)$, then $p(\mathbf{w})$ is also a Gaussian with shifted mean $\sum_{i,\mathbf{y}} \alpha_i(\mathbf{y})\Delta \mathbf{f}_i(\mathbf{y})$ and covariance matrix $I$, where the Lagrangian multipliers $\alpha_i(\mathbf{y})$ can be obtained by solving problem D1 of the form that is isomorphic to the dual of M³N. When applying this $p(\mathbf{w})$ to Eq. (3), one can obtain a predictor that is identical to that of the M³N.

From the above reduction, it should be clear that M³N is a special case of MaxEnDNet. But the MaxEnDNet in its full generality offers a number of important advantages while retaining all the

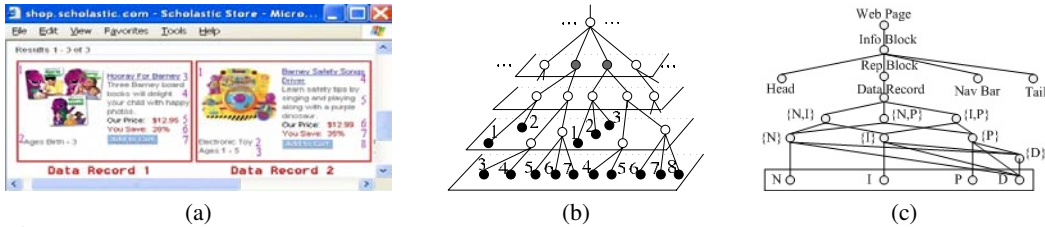

| (a) | (b) | (c) |

Figure 1: (a) A web page with two data records containing 7 and 8 elements respectively; (b) A partial vision tree of the page in Figure 1(a), where grey nodes are the roots of the two records; (c) A label hierarchy for product information extraction, where the root node represents an entire instance (a web page); leaf nodes are the attributes (i.e. *N*ame, *I*mage, *P*rice, and *D*escription); and inner nodes are the intermediate class labels defined for parts of a web page, e.g. {N, I} is a class label for blocks containing both *N*ame and *I*mage.

merits of the M$^3$N. **First**, the MaxEnDNet prediction is based on model averaging and therefore enjoys a desirable smoothing effect, with a uniform convergence bound on generalization error, as shown in [20]. **Second**, MaxEnDNet admits a prior that can be designed to introduce useful regularization effects, such as a sparsity bias, as explored in the Laplace M$^3$N [19, 20]. **Third**, as explored in this paper, MaxEnDNet offers a principled way to incorporate *hidden* generative models underlying the structured predictions, but allows the predictive model to be discriminatively trained based on partially labeled data. In the sequel, we introduce *partially observed* MaxEnDNet (PoMEN), that combines (possibly latent) generative model and discriminative training for structured prediction.

## 3 Partially Observed MaxEnDNet

Consider, for example, the problem of web data extraction, which is to identify interested information from web pages. Each sample is a data record or an entire web page which is represented as a set of HTML elements. One striking characteristic of web data extraction is that various types of structural dependencies between HTML elements exist, e.g. the HTML tag tree or the Document Object Model (DOM) structure is itself hierarchical. In [17], fully observed hierarchical CRFs are shown to have great promise and achieve better performance than flat models like linear-chain CRFs [7]. One method to construct a hierarchical model is to first use a parser to construct a so called vision tree [17]. For example, Figure 1(b) is a part of the vision tree of the page in Figure 1(a). Then, based on the vision tree, a hierarchical model can be constructed accordingly to extract the interested attributes, e.g. a product's name, image, price, description, etc. In such a hierarchical extraction model, inner nodes are useful to incorporate long distance dependencies, and the variables at one level are refinements of the variables at upper levels. To reflect the refinement relationship, the class labels defined as in [17] are also organized in a hierarchy as in Figure 1(c). Due to concerns over labeling cost and annotation-ambiguity caused by the overlapping of class labels as in Figure 1(c), it is desirable to effectively learn a hierarchical extraction model with partially labeled data.

Without loss of generality, assume that the structured labeling of a sample consists of two parts—an observed part $\mathbf{y}$ and a hidden part $\mathbf{z}$. Both $\mathbf{y}$ and $\mathbf{z}$ are structured labels, and furthermore the hidden variables are not isolated, but are statistically dependent on each other and on the observed data according to a graphical model $p(\mathbf{y}, \mathbf{z}, \mathbf{w}|\mathbf{x}) = p(\mathbf{w}, \mathbf{z}|\mathbf{x})p(\mathbf{y}|\mathbf{x}, \mathbf{z}, \mathbf{w})$, where $p(\mathbf{y}|\mathbf{x}, \mathbf{z}, \mathbf{w})$ takes the form of a Boltzmann distribution $p(\mathbf{y}|\mathbf{x}, \mathbf{z}, \mathbf{w}) = \frac{1}{Z} \exp\{-F(\mathbf{x}, \mathbf{y}, \mathbf{z}; \mathbf{w})\}$ and $\mathbf{x}$ is a global condition as in CRFs [7]. Following the spirit of a margin-based structured predictor such as M$^3$N, we employ only the unnormalized energy function $F(\mathbf{x}, \mathbf{y}, \mathbf{z}; \mathbf{w})$ (which usually consists of linear combinations of feature functions or potentials) as the cost function for structured prediction, and we adopt a prediction rule directly extended from the MaxEnDNet—average over all the possible models defined by different $\mathbf{w}$, and at the same time marginalized over all hidden variables $\mathbf{z}$. That is,

$$h_2(\mathbf{x}) = \arg \max_{\mathbf{y} \in \mathcal{Y}(\mathbf{x})} \sum_{\mathbf{z}} \int p(\mathbf{w}, \mathbf{z}) F(\mathbf{x}, \mathbf{y}, \mathbf{z}; \mathbf{w}) \, d\mathbf{w} \,. \qquad (6)$$

Now our problem is learning the optimum $p(\mathbf{w}, \mathbf{z})$ from data. Let $\{\mathbf{z}\} \equiv (\mathbf{z}^1, \ldots, \mathbf{z}^N)$ denote the ensemble of hidden labels of all the samples. Analogous to the setup for learning the MaxEnDNet, we specify a prior distribution $p_0(\{\mathbf{z}\})$ over all the hidden structured labels. The feasible space $\mathcal{F}_2$ of $p(\mathbf{w}, \{\mathbf{z}\})$ can be defined as follows according to the margin constraints:

$$\mathcal{F}_2 = \Big\{ p(\mathbf{w}, \{\mathbf{z}\}) : \sum_{\mathbf{z}} \int p(\mathbf{w}, \mathbf{z})[\Delta F_i(\mathbf{y}, \mathbf{z}; \mathbf{w}) - \Delta \ell_i(\mathbf{y})] \, d\mathbf{w} \geq -\xi_i, \, \forall i, \, \forall \mathbf{y} \Big\},$$

where $\Delta F_i(\mathbf{y}, \mathbf{z}; \mathbf{w}) = F(\mathbf{x}^i, \mathbf{y}^i, \mathbf{z}; \mathbf{w}) - F(\mathbf{x}^i, \mathbf{y}, \mathbf{z}; \mathbf{w})$, and $p(\mathbf{w}, \mathbf{z})$ is the marginal distribution of $p(\mathbf{w}, \{\mathbf{z}\})$ on a single sample, which will be used in (6) to compute the structured prediction.

Again we learn the optimum $p(\mathbf{w}, \{\mathbf{z}\})$ based on a *structured minimum relative entropy principle* as in MaxEnDNet. Specifically, let $p_0(\mathbf{w}, \{\mathbf{z}\})$ represent a given joint prior over the parameters and the hidden variables, we define the PoMEN problem that gives rise to the optimum $p(\mathbf{w}, \{\mathbf{z}\})$:

$$\text{P2 (PoMEN)}: \quad \min_{p(\mathbf{w},\{\mathbf{z}\})\in\mathcal{F}_2,\xi\in\mathbb{R}_+^N} KL(p(\mathbf{w},\{\mathbf{z}\})||p_0(\mathbf{w},\{\mathbf{z}\})) + U(\xi). \tag{7}$$

Analogous to P1, P2 is a variational optimization problem over $p(\mathbf{w}, \{\mathbf{z}\})$ in the feasible space $\mathcal{F}_2$. Again since both the KL and the $U$ function in P2 are convex, and the constraints in $\mathcal{F}_2$ are linear, P2 is a convex program. Thus, we can employ a technique similar to that used to solve MaxEnDNet to solve the PoMEN problem.

### 3.1 Learning PoMEN

For a fully general $p(\mathbf{w}, \{\mathbf{z}\})$ where hidden variables in all samples are coupled, solving P2 based on an extension of Theorem 1 would involve very high-dimensional integration and summation that is in practice intractable. In this paper we consider a simpler case where the hidden labels of different samples are *iid* and independent of the parameter $\mathbf{w}$ in both the prior and the posterior distributions, that is, $p_0(\mathbf{w}, \{\mathbf{z}\}) = p_0(\mathbf{w}) \prod_{i=1}^N p_0(\mathbf{z}^i)$ and $p(\mathbf{w}, \{\mathbf{z}\}) = p(\mathbf{w}) \prod_{i=1}^N p(\mathbf{z}^i)$. This assumption will hold true in a graphical model where $\mathbf{w}$ corresponds to only the observed $\mathbf{y}$ variables at the bottom of a hierarchical model. For many practical applications such as the hierarchical web-info extraction, such a model is realistic and adequate. For more general models where dependencies are more global, we can use the above factored model as a generalized mean field approximation to the true distribution, but this extension is beyond the scope of this paper, and will be explored later in the full paper. Generalizing Theorem 1, following a coordinate descent principle, now we present an alternating minimization (EM-style) procedure for P2:

**Step 1**: keep $p(\mathbf{z})$ fixed, infer $p(\mathbf{w})$ by solving the following problem:

$$\min_{p(\mathbf{w})\in\mathcal{F}_1',\xi\in\mathbb{R}_+^N} KL(p(\mathbf{w})||p_0(\mathbf{w})) + C\sum_i \xi_i, \tag{8}$$

where $\mathcal{F}_1' = \{p(\mathbf{w}): \int p(\mathbf{w})E_{p(\mathbf{z})}[\Delta F_i(\mathbf{y}, \mathbf{z}; \mathbf{w}) - \Delta\ell_i(\mathbf{y})]\,d\mathbf{w} \geq -\xi_i, \forall i, \forall\mathbf{y}\}$, which is a generalized version of $\mathcal{F}_1$ with hidden variables. Thus, we can apply the same convex optimization techniques as being used for solving the problem P1. Specifically, assume that the prior distribution $p_0(\mathbf{w})$ is a standard normal and $F(\mathbf{x}, \mathbf{y}, \mathbf{z}; \mathbf{w}) = \mathbf{w}^\top \mathbf{f}(\mathbf{x}, \mathbf{y}, \mathbf{z})$, then the solution (i.e. posterior distribution) is $p(\mathbf{w}) = \mathcal{N}(\mathbf{w}|\mu_\mathbf{w}, I)$, where $\mu_\mathbf{w} = \sum_{i,\mathbf{y}} \alpha_i(\mathbf{y})E_{p(\mathbf{z})}[\Delta\mathbf{f}_i(\mathbf{y}, \mathbf{z})]$. The dual variables $\alpha$ are achieved by solving a dual problem:

$$\max_{\alpha\in\mathcal{P}(C)} \sum_{i,\mathbf{y}} \alpha_i(\mathbf{y})\Delta\ell_i(\mathbf{y}) - \frac{1}{2}\|\sum_{i,\mathbf{y}} \alpha_i(\mathbf{y})E_{p(\mathbf{z})}[\Delta\mathbf{f}_i(\mathbf{y}, \mathbf{z})]\|^2, \tag{9}$$

where $\mathcal{P}(C) = \{\alpha: \sum_\mathbf{y} \alpha_i(\mathbf{y}) = C; \alpha_i(\mathbf{y}) \geq 0, \forall i, \forall\mathbf{y}\}$. This dual problem is isomorphic to the dual form of the $M^3N$ optimization problem, and we can use existing algorithms developed for $M^3N$, such as [12, 3] to solve it. Alternatively, we can solve the following primal problem via employing existing subgradient [10] or cutting plane [13] algorithms:

$$\min_{\mathbf{w}\in\mathcal{F}_0',\xi\in\mathbb{R}_+^N} \frac{1}{2}\mathbf{w}^\top\mathbf{w} + C\sum_{i=1}^N \xi_i, \tag{10}$$

where $\mathcal{F}_0' = \{\mathbf{w}: \mathbf{w}^\top E_{p(\mathbf{z})}[\Delta\mathbf{f}_i(\mathbf{y}, \mathbf{z})] \geq \Delta\ell_i(\mathbf{y}) - \xi_i; \xi_i \geq 0, \forall i, \forall\mathbf{y}\}$, which is a generalized version of $\mathcal{F}_0$. It is easy to show that the solution to this primal problem is the posterior mean of $p(\mathbf{w})$, which will be used to make prediction in the predictive function $h_2$. Note that the primal problem is very similar to that of $M^3N$, except the expectations in $\mathcal{F}_0'$. This is not surprising since it can be shown that $M^3N$ is a special case of MaxEnDNet. We will discuss how to efficiently compute the expectations $E_{p(\mathbf{z})}[\Delta\mathbf{f}_i(\mathbf{y}, \mathbf{z})]$ in Step 2.

**Step 2**: keep $p(\mathbf{w})$ fixed, based on the factorization assumption $p(\{\mathbf{z}\}) = \prod_i p(\mathbf{z}^i)$ and $p_0(\{\mathbf{z}\}) = \prod_i p_0(\mathbf{z}^i)$, the distribution $p(\mathbf{z})$ for each sample $i$ can be obtained by solving the following problem:

$$\min_{p(\mathbf{z})\in\mathcal{F}_1^\star,\xi_i\in\mathbb{R}_+} KL(p(\mathbf{z})||p_0(\mathbf{z})) + C\xi_i, \tag{11}$$

where $\mathcal{F}_1^\star = \{p(\mathbf{z}) : \sum_{\mathbf{z}} p(\mathbf{z}) \int p(\mathbf{w})[\mathbf{w}^\top \Delta \mathbf{f}_i(\mathbf{y}, \mathbf{z}) - \Delta \ell_i(\mathbf{y})] \, d\mathbf{w} \geq -\xi_i, \ \forall \mathbf{y}\}$. Since $p(\mathbf{w})$ is a normal distribution as shown in Step 1, $\mathcal{F}_1^\star = \{p(\mathbf{z}) : \sum_{\mathbf{z}} p(\mathbf{z})[\mu_{\mathbf{w}}^\top \Delta \mathbf{f}_i(\mathbf{y}, \mathbf{z}) - \Delta \ell_i(\mathbf{y})] \geq -\xi_i, \ \forall \mathbf{y}\}$. Similarly, by introducing a set of Lagrangian multipliers $\beta(\mathbf{y})$, we can get:

$$p(\mathbf{z}) = \frac{1}{Z(\beta)} p_0(\mathbf{z}) \exp \Big\{ \sum_{\mathbf{y}} \beta(\mathbf{y})[\mu_{\mathbf{w}}^\top \Delta \mathbf{f}_i(\mathbf{y}, \mathbf{z}) - \Delta \ell_i(\mathbf{y})] \Big\},$$

and the dual variables $\beta(\mathbf{y})$ can be obtained by solving the following dual problem:

$$\max_{\beta \in \mathcal{P}_i(C)} - \log \Big( \sum_{\mathbf{z}} p_0(\mathbf{z}) \exp\{ \sum_{\mathbf{y}} \beta(\mathbf{y})[\mu_{\mathbf{w}}^\top \Delta \mathbf{f}_i(\mathbf{y}, \mathbf{z}) - \Delta \ell_i(\mathbf{y})]\} \Big), \qquad (12)$$

where $\mathcal{P}_i(C) = \{\sum_{\mathbf{y}} \beta(\mathbf{y}) = C, \ \beta(\mathbf{y}) \geq 0, \ \forall \mathbf{y}\}$. This non-linear constrained optimization problem can be solved with existing solvers, like IPOPT [15]. With a little algebra, we can compute the gradients as follows:

$$\frac{\partial \log Z(\beta)}{\partial \beta(\mathbf{y})} = \mu_{\mathbf{w}}^\top E_{p(\mathbf{z})}[\Delta \mathbf{f}_i(\mathbf{y}, \mathbf{z})] - \Delta \ell_i(\mathbf{y}).$$

To efficiently calculate the expectations $E_{p(\mathbf{z})}[\Delta \mathbf{f}_i(\mathbf{y}, \mathbf{z})]$ as required in Step1 and in the above gradients. We make a gentle assumption that the prior distribution $p_0(\mathbf{z})$ is an exponential distribution of the following form:

$$p_0(\mathbf{z}) = \exp \Big\{ \sum_m \phi_m(\mathbf{z}) \Big\}. \qquad (13)$$

This assumption is general enough for our purpose, and covers the following commonly used priors:

i. **Log-linear Prior**: defined by a set of feature functions and their weights. For example, in a pairwise Markov network, we can define the prior model as: $p_0(\mathbf{z}) \propto \exp \big\{ \sum_{(i,j) \in E} \sum_k \lambda_k g_k(z_i, z_j) \big\}$, where $g_k(z_i, z_j)$ are feature functions and $\lambda_k$ are weights.

ii. **Independent Prior**: defined as $p_0(\mathbf{z}) = \prod_{j=1}^{\ell} p_0(z_j)$. In the logarithm space, we can write it as: $p_0(\mathbf{z}) = \exp\{\sum_{j=1}^{\ell} \log p_0(z_j)\}$.

iii. **Markov Prior**: the prior model have the Markov property w.r.t the model's structure. For example, for a chain graph, the prior distribution can be written as: $p_0(\mathbf{z}) = p(z_1) \prod_{j=2}^{\ell} p_0(z_j | z_{j-1})$. Similarly, in the logarithm space, $p_0(\mathbf{z}) = \exp\{\log p_0(z_1) + \sum_{j=2}^{\ell} \log p_0(z_j | z_{j-1})\}$.

With the above assumption, $p(\mathbf{z})$ is an exponential family distribution, and the expectations, $E_{p(\mathbf{z})}[\Delta \mathbf{f}_i(\mathbf{y}, \mathbf{z})]$, can be efficiently calculated by exploring the sparseness of the model's structure to compute marginal probabilities, e.g. $p(z_i)$ and $p(z_i, z_j)$ in pairwise Markov networks. When the model's tree width is not large, this can be done exactly. For complex models, approximate inference like loopy belief propagation and variational methods can be applied. However, since the number of constraints in (12) is exponential to the size of the observed labels, the optimization problem cannot be efficiently solved. A key observation, as explored in [12], is that we can interpret $\beta(\mathbf{y})$ as a probability distribution of $\mathbf{y}$ because of the regularity constraints: $\sum_{\mathbf{y}} \beta(\mathbf{y}) = C, \ \beta(\mathbf{y}) \geq 0, \ \forall \mathbf{y}$. Thus, we can introduce a set of marginal dual variables and transfer the dual problem (12) to an equivalent form with a polynomial number of constraints. The derivatives with respect to each marginal dual parameter is of the same structure as the above gradients.

## 4 Experiments

We apply PoMEN to the problem of web data extraction, and compare it with partially observed CRFs (PoHCRF) [9], and fully observed hierarchical CRFs (HCRF) [17] and hierarchical M³N (HM³N) which has the same hierarchical model structure as the HCRF.

### 4.1 Data Sets, Evaluation Criteria, and Prior for Latent Variables

We concern ourselves with the problem of identifying product items for sale on the web. For each product item, four attributes – *Name*, *Image*, *Price*, and *Description* are extracted in our experiments. The evaluation data consists of product web pages generated from 37 different templates. For each template, there are 5 pages for training and 10 for testing. We evaluate all the methods on two different levels of inputs, *record level* and *page level*. For record-level evaluation, we assume that data records are given, and we compare different models on accuracy of extracting attributes in the given records. For page-level evaluation, the inputs are raw web pages and all the models perform

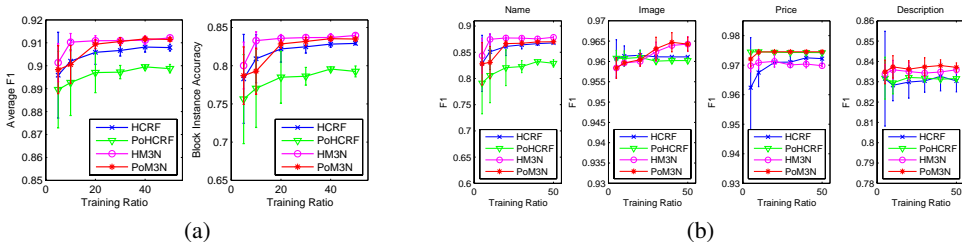

Figure 2: (a) The F1 and block instance accuracy of record-level evaluation from 4 models under different amount of training data. (b) The F1 and its variance on the attributes: *Name*, *Image*, *Price*, and *Description*.

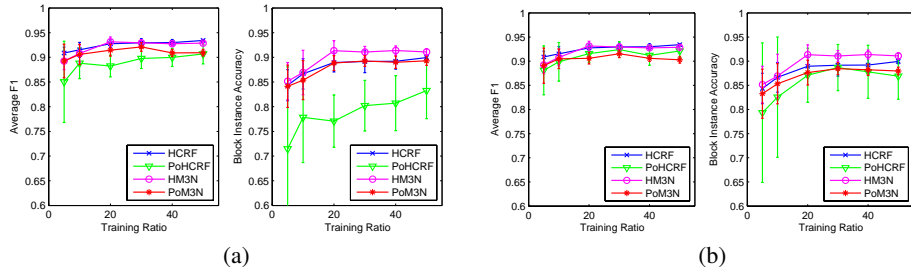

Figure 3: The average F1 and block instance accuracy of different models with different ratios of training data for two types of page-level evaluation: (a) **ST1**; and (b) **ST2**.

both record detection and attribute extraction simultaneously as in [17]. In the 185 training pages, there are 1585 data records in total; in the 370 testing pages, 3391 data records are collected. As for evaluation criteria, we use the standard precision, recall, and their harmonic value F1 for each attribute and the two comprehensive measures, i.e. average F1 and block instance accuracy, as defined in [17]. We adopt an independent prior described earlier for the latent variables, each factor $p_0(z_i)$ over a single latent label is assumed to be uniform.

### 4.2 Record-Level Evaluation

In this evaluation, partially observed training data are the data records whose leaf nodes are labeled and inner nodes are hidden. We randomly select $m = 5, 10, 20, 30, 40,$ or, $50$ percent of the training records as training data, and test on all the testing records. For each $m$, 10 independent experiments were conducted and the average performance is summarized in Figure 2. From Figure 2(a), it can be seen that the HM³N performs slightly better than HCRF trained on fully labeled data. For the two partially observed models, PoMEN performs much better than PoHCRF in both average F1 and block instance accuracy, and with lower variances of the score, especially when the training set is small. As the number of training data increases, PoMEN performs comparably w.r.t. the fully observed HM³N. For all the models, higher scores and lower variances are achieved with more training data. Figure 2(b) shows the F1 score on each attribute. Overall, for attributes *Image*, *Price*, and *Description*, although all models generally perform better with more training data, the improvement is small; and the differences between different models are small. This is possibly because the features of these attributes are usually consistent and distinctive, and therefore easier to learn and predict. For the attribute *Name*, however, a large number of training data are needed to learn a good model because its underlying features have diverse appearance on web pages.

### 4.3 Page-Level Evaluation

Experiments on page-level prediction is conducted similarly as above, and the results are summarized in Figure 3. Two different partial labeling strategies are used to generate training data. **ST1**: label the leaf nodes and the nodes that represent data records; **ST2**: label more information based on ST1, e.g., label also the nodes above the "Data Record" nodes in the hierarchy as in Figure 1(c). Due to space limitation, we only report average F1 and block instance accuracy.

For **ST1**, PoMEN achieves better scores and lower variances than PoHCRF in both average F1 and block instance accuracy. The HM³N performs slightly better than HCRF (both trained on full labeling), and PoMEN performs comparably with the fully observed HCRF in block instance accuracy. For **ST2**, with more supervision information, PoHCRF achieves higher performance that is comparable to that of HM³N in average F1, but slightly lower than HM³N in block instance accuracy. For

the latent models, PoHCRF performs slightly better in average F1, and PoMEN does better in block instance accuracy; moreover, the variances of PoMEN are much smaller than those of PoHCRF in both average F1 and block instance accuracy. We can also see that PoMEN does not change much when additional label information is provided in **ST2**. Thus, the max-margin principle could provide a better paradigm than the likelihood-based estimation for learning latent hierarchical models.

For the second step of learning PoMEN, the IPOPT solver [15] was used to compute the distribution $p(\mathbf{z})$. Interestingly, the performance of PoMEN does not change much during the iteration, and our results were achieved within 3 iterations. It is possible that in hierarchical models, since inner variables usually represent overlapping concepts, the initial distribution are already reasonably good to describe confidence on the labeling due to implicit consistence across the labels. This is unlike the multi-label learning [6] where only one of the multiple labels is true and during the iteration more probability mass should be redistributed on the true label during the EM iterations.

## 5 Conclusions

We have presented an extension of the standard max-margin learning to address the challenging problem of learning Markov networks with the existence of structured hidden variables. Our approach is a generalization of the maximum entropy discrimination Markov networks (MaxEnDNet), which offer a general framework to combine Bayesian-style and max-margin learning and subsume the standard M$^3$N as a special case, to consider structured hidden variables. For the partially observed MaxEnDNet, we developed an EM-style algorithm based on existing convex optimization algorithms developed for the standard M$^3$N. We applied the proposed model to a real-world web data extraction task and showed that learning latent hierarchical models based on the max-margin principle could be better than the likelihood-based learning with hidden variables.

### Acknowledgments

This work was done while J.Z. was a visiting researcher at CMU under a State Scholarship from China, and supports from NSF DBI-0546594 and DBI-0640543 awarded to E.X.; J.Z. and B.Z. are also supported by Chinese NSF Grant 60621062 and 60605003; National Key Foundation R&D Projects 2003CB317007, 2004CB318108 and 2007CB311003; and Basic Research Foundation of Tsinghua National Lab for Info Sci & Tech.

## References

[1] Y. Altun, D. McAllester, and M. Belkin. Maximum margin semi-supervised learning for structured variables. In *NIPS*, 2006.

[2] Y. Altun, I. Tsochantaridis, and T. Hofmann. Hidden markov support vector machines. In *ICML*, 2003.

[3] P. Bartlett, M. Collins, B. Taskar, and D. McAllester. Exponentiated gradient algorithms for larg-margin structured classification. In *NIPS*, 2004.

[4] U. Brefeld and T. Scheffer. Semi-supervised learning for structured output variables. In *ICML*, 2006.

[5] M. Dudík, S.J. Phillips, and R.E. Schapire. Maximum entropy density estimation with generalized regularization and an application to species distribution modeling. *JMLR*, (8):1217–1260, 2007.

[6] R. Jin and Z. Ghahramani. Learning with multiple labels. In *NIPS*, 2002.

[7] J. Lafferty, A. McCallum, and F. Pereira. Conditional random fields: Probabilistic models for segmenting and labeling sequence data. In *ICML*, 2001.

[8] G. Lebanon and J. Lafferty. Boosting and maximum likelihood for exponential models. In *NIPS*, 2001.

[9] A. Quattoni, M. Collins, and T. Darrell. Conditional random fields for object recognition. In *NIPS*, 2004.

[10] N.D. Ratliff, J.A. Bagnell, and M.A. Zinkevich. (online) subgradient methods for structured prediction. In *AISTATS*, 2007.

[11] F. Sha and L. Saul. Large margin hidden markov models for automatic speech recognition. In *NIPS*, 2006.

[12] B. Taskar, C. Guestrin, and D. Koller. Max-margin markov networks. In *NIPS*, 2003.

[13] I. Tsochantaridis, T. Hofmann, T. Joachims, and Y. Altun. Support vector machine learning for interdependent and structured output spaces. In *ICML*, 2004.

[14] J. Verbeek and B. Triggs. Scene segmentation with conditional random fields learned from partially labeled images. In *NIPS*, 2007.

[15] A. Wächter and L.T. Biegler. On the implementation of a primal-dual interior point filter line search algorithm for large-scale nonlinear programming. *Mathematical Programming*, (106(1)):25–57, 2006.

[16] L. Xu, D. Wilkinson, F. Southey, and D. Schuurmans. Discriminative unsupervised learning of structured predictors. In *ICML*, 2006.

[17] J. Zhu, Z. Nie, J.-R. Wen, B. Zhang, and W.-Y. Ma. Simultaneous record detection and attribute labeling in web data extraction. In *SIGKDD*, 2006.

[18] J. Zhu, Z. Nie, B. Zhang, and J.-R. Wen. Dynamic hierarchical markov random fields and their application to web data extraction. In *ICML*, 2007.

[19] J. Zhu, E.P. Xing, and B. Zhang. Laplace maximum margin markov networks. In *ICML*, 2008.

[20] J. Zhu, E.P. Xing, and B. Zhang. Maximum entropy discrimination markov networks. Technical Report CMU-ML-08-104, Machine Learning Department, Carnegie Mellon University, 2008.

